# Self Supervised Boosting

**Max Welling, Richard S. Zemel, and Geoffrey E. Hinton**
Department of Computer Science
University of Toronto
10 King's College Road
Toronto, M5S 3G5 Canada

## Abstract

Boosting algorithms and successful applications thereof abound for classification and regression learning problems, but not for unsupervised learning. We propose a sequential approach to adding features to a random field model by training them to improve classification performance between the data and an equal-sized sample of "negative examples" generated from the model's current estimate of the data density. Training in each boosting round proceeds in three stages: first we sample negative examples from the model's current Boltzmann distribution. Next, a feature is trained to improve classification performance between data and negative examples. Finally, a coefficient is learned which determines the importance of this feature relative to ones already in the pool. Negative examples only need to be generated once to learn each new feature. The validity of the approach is demonstrated on binary digits and continuous synthetic data.

## 1 Introduction

While researchers have developed and successfully applied a myriad of boosting algorithms for classification and regression problems, boosting for density estimation has received relatively scant attention. Yet incremental, stage-wise fitting is an attractive model for density estimation. One can imagine that the initial *features*, or *weak learners*, could model the rough outlines of the data density, and more detailed carving of the density landscape could occur on each successive round. Ideally, the algorithm would achieve automatic model selection, determining the requisite number of weak learners on its own. It has proven difficult to formulate an objective for such a system, under which the weights on examples, and the objective for training a weak learner at each round have a natural gradient-descent interpretation as in standard boosting algorithms [10] [7]. In this paper we propose an algorithm that provides some progress towards this goal.

A key idea in our algorithm is that unsupervised learning can be converted into supervised learning by using the model's imperfect current estimate of the data to generate negative examples. A form of this idea was previously exploited in the *contrastive divergence* algorithm [4]. We take the idea a step further here by training a weak learner to discriminate between the positive examples from the original data and the negative examples generated by sampling from the current density estimate. This new weak learner minimizes a simple additive logistic loss function [2].

Our algorithm obtains an important advantage over sampling-based, unsupervised methods that learn features in parallel. Parallel-update methods require a new sample after each iteration of parameter changes, in order to reflect the current model's estimate of the data density. We improve on this by using one sample per boosting round, to fit one weak learner. The justification for this approach comes from the proposal that, for stagewise additive models, boosting can be considered as gradient-descent in function space, so the new learner can simply optimize its inner product with the gradient of the objective in function space [3].

Unlike other attempts at "unsupervised boosting" [9], where at each round a new component distribution is added to a *mixture* model, our approach will add features in the log-domain and as such learns a *product model*.

Our algorithm incrementally constructs random fields from examples. As such, it bears some relation to maximum entropy models, which are popular in natural language processing [8]. In these applications, the features are typically not learned; instead the algorithms greedily select at each round the most informative feature from a large set of pre-enumerated features.

## 2   The Model

Let the input, or *state* $\mathbf{s}$ be a vector of $D$ random variables taking values in some finite domain $\mathbb{S}^D$. The probability of $\mathbf{s}$ is defined by assigning it an *energy*, $E(\mathbf{s})$, which is converted into a probability using the Boltzmann distribution,

$$P(\mathbf{s}) = \frac{1}{Z} \exp\left[-E(\mathbf{s})\right] \qquad Z = \sum_{\mathbf{s}} \exp\left[-E(\mathbf{s})\right] \tag{1}$$

We furthermore assume that the energy is *additive*. More explicitly, it will be modelled as a weighted sum of features,

$$E(\mathbf{s}) = \sum_t E_t(\mathbf{s}) = \sum_t \alpha_t \phi_t(\mathbf{s}; \theta_t) \tag{2}$$

where $\{\alpha_t\}$ are the weights, $\{\phi_t(\cdot)\}$ the features and each feature may depend on its own set of parameters $\theta_t$.

The model described above is very similar to an "additive random field", otherwise known as "maximum entropy model". The key difference is that we allow each feature to be flexible through its dependence on the parameters $\theta_t$.

Learning in random fields may proceed by performing gradient ascent on the log-likelihood:

$$\frac{\partial L}{\partial \beta} = -\frac{1}{M} \sum_{m=1}^{M} \frac{\partial E(\mathbf{d}_m)}{\partial \beta} + \sum_{\mathbf{s} \in \mathbb{S}^D} P(\mathbf{s}) \frac{\partial E(\mathbf{s})}{\partial \beta} \tag{3}$$

where $\mathbf{d}_m$ is a data-vector and $\beta$ is some arbitrary parameter that we want to learn. This equation makes explicit the main philosophy behind learning in random fields: *the energy of states "occupied" by data is lowered (weighted by $\frac{1}{M}$) while the energy of* **all** *states is raised (weighted by $P(\mathbf{s})$).* Since there are usually an exponential number of states in the system, the second term is often approximated by a sample from $P(\mathbf{s})$. To reduce sampling noise a relatively large sample is necessary and moreover, it must be drawn each time we compute gradients. These considerations make learning in random fields generally very inefficient.

Iterative scaling methods have been developed for models that do not include adaptive feature parameters $\{\theta_t\}$ but instead train only the coefficients $\{\alpha_t\}$ [8]. These methods make more efficient use of the samples than gradient ascent, but they only minimize a loose bound on the cost function and their terminal convergence can be slow.

# 3 An Algorithm for Self Supervised Boosting

Boosting algorithms typically implement 3 phases: a feature (or weak learner) is trained, the relative weight of this feature with respect to the other features already in the pool is determined, and finally the data vectors are reweighted. In the following we will discuss a similar strategy in an unsupervised setting.

## 3.1 Finding New Features

In [7], boosting is reinterpreted as *functional gradient descent* on a loss function. Using the log-likelihood as a negative loss function this idea can be used to find features for additive random field models. Consider a change in the energy by adding an infinitesimal multiple of a feature. The optimal feature is then the one that provides the maximal increase in log-likelihood, i.e. the feature that maximizes the second term of

$$L\left[E(\mathbf{s}) + \varepsilon\phi_t(\mathbf{s})\right] \approx L\left[E(\mathbf{s})\right] + \varepsilon \left.\frac{\partial L}{\partial\varepsilon}\right|_{\varepsilon=0} \tag{4}$$

Using Eqn. 3 with $\partial E/\partial\varepsilon = \phi_t$ we rewrite the second term as,

$$\left.\frac{\partial L}{\partial\varepsilon}\right|_{\varepsilon=0} = -\frac{1}{M}\sum_{m=1}^{M}\phi_t(\mathbf{d}_m) + \sum_{\mathbf{s}\in\mathbb{S}^D}P(\mathbf{s})\phi_t(\mathbf{s}) \tag{5}$$

where $P(\mathbf{s})$ is our current estimate of the data distribution. In order to maximize this derivative, the feature should therefore be small at the data and large at all other states. It is however important to realize that the norm of the feature must be bounded, since otherwise the derivative can be made arbitrarily large by simply increasing the length of $\phi_t(\mathbf{s})$.

Because the total number of possible states of a model is often exponentially large, the second term of Eqn. 5 must be approximated using samples $\mathbf{s}_n$ from $P(\mathbf{s})$,

$$\left.\frac{\partial L}{\partial\varepsilon}\right|_{\varepsilon=0} \approx -\frac{1}{M}\sum_{m=1}^{M}\phi_t(\mathbf{d}_m) + \frac{1}{N}\sum_{n=1}^{N}\phi_t(\mathbf{s}_n) \tag{6}$$

These samples, or "negative examples", inform us about the states that are likely under the current model. Intuitively, because the model is imperfect, we would like to move its density estimate away from these samples and towards the actual data. By labelling the data with $y = -1$ and the negative examples with $y = +1$, we can map this to a supervised problem where a new feature is a classifier. Since a good classifier is negative at the data and positive at the negative examples (so we can use its sign to discriminate them), *adding* its output to the total energy will lower the energy at states where there are data and raise it at states where there are negative examples. The main difference with supervised boosting is that the negative examples change at every round.

## 3.2 Weighting the Data

It has been observed [6] that boosting algorithms can outperform classifications algorithms that maximize log-likelihood. This has motivated us to use the *logistic loss function* from the boosting literature for training new features.

$$\text{Loss} = \sum_{k}\log\left(1 + e^{-y_k E_k}\right) \tag{7}$$

where $k$ runs over data ($y_k = -1$) and negative examples ($y_k = +1$). Perturbing the energy of the negative loss function by adding an infinitesimal multiple of a new feature:

$E \rightarrow E + \varepsilon\phi_t$ and computing the derivative w.r.t. $\varepsilon$ we derive the following cost function for adding a new feature,

$$\mathcal{C} = -\sum_{m=1}^{M} w_m \phi_t(\mathbf{d}_m) + \sum_{n=1}^{N} w_n \phi_t(\mathbf{s}_n) \qquad w_k = \sigma(-y_k E_k) \qquad (8)$$

The main difference with Eqn. 6 is the weights $w_k$ on data and negative examples, that give poorly "classified" examples (data with very high energy and negative examples with very low energy) a stronger vote in changes to the energy surface. The extra weights (which are bounded between [0,1]) will incur a certain bias w.r.t. the maximum likelihood solution. However, it is expected that the extra effort on "hard cases" will cause the algorithm to converge faster to good density models.

It is important to realize that the loss function Eqn. 7 is a valid cost function *only* when the negative examples are fixed. The reason is that after a change of the energy surface, the negative examples are no longer a representative sample from the Boltzmann distribution in Eqn. 1. However, as long as we re-sample the negative examples after every change in the energy we may use Eqn. 8 as an objective to decide what feature to add to the energy, i.e. we may consider it as the derivative of some (possibly unknown) weighted log-likelihood: $\mathcal{C} = \partial L_w / \partial \varepsilon|_{\varepsilon=0}$.

By analogy, we can interpret $p(y = -1|\mathbf{s}) = \sigma(-E(\mathbf{s}))$ as the probability that a certain state $\mathbf{s}$ is occupied by a data-vector and consequently $-E(\mathbf{s})$ as the "margin". Note that the introduction of the weights has given meaning to the "height" of the energy surface, in contrast to the Boltzmann distribution for which only relative energy differences count. In fact, as we will further explain in the next section, the height of the energy will be chosen such that the total weight on data is equal to the total weight on the negative examples.

### 3.3 Adding the New Feature to the Pool

According to the functional gradient interpretation, the new feature computed as described above represents the infinitesimal change in energy that maximally increases the (weighted) log-likelihood. Consistent with that interpretation we will determine $\alpha_t$ via a line search in the direction of this "gradient". In fact, we will propose a slightly more general change in energy given by,

$$E(\mathbf{s}) \rightarrow E(\mathbf{s}) + \alpha_t \phi_t(\mathbf{s}) + \gamma_t \qquad (9)$$

As mentioned in the previous section, the constant $\gamma_t$ will have no effect on the Boltzmann distribution in Eqn. 1. However, it does influence the relative total weight on data versus negative examples. Using the interpretation of $\mathcal{C}$ in Eqn. 8 as $\partial L_w / \partial \varepsilon|_{\varepsilon=0}$ it is not hard to see that the derivatives of $L_w$ w.r.t. to $\alpha_t$ and[1] $\gamma_t$ are given by,

$$\frac{\partial L_w}{\partial \alpha_t} = -\sum_{m=1}^{M} w_m \phi_t(\mathbf{d}_m) + \sum_{n=1}^{N} w_n \phi_t(\mathbf{s}_n) \qquad (10)$$

$$\frac{\partial L_w}{\partial \gamma_t} = -\sum_{m=1}^{M} w_m + \sum_{n=1}^{N} w_n \qquad (11)$$

Therefore, at a stationary point of $L_w$ w.r.t. $\gamma_t$ the total weight on data and negative examples precisely balances out.

When iteratively updating $\alpha_t$ we not only change the weights $w_k$ but also the Boltzmann distribution, which makes the negative examples no longer representative of the current

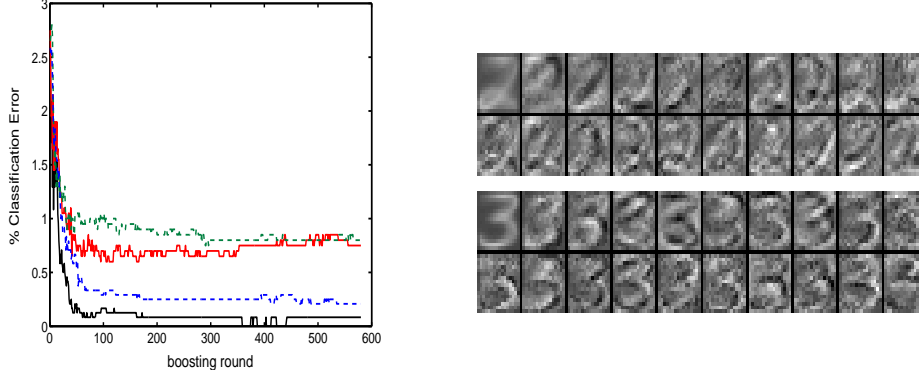

Figure 1: (a – left). Training error (lower 2 curves) and test error (higher 2 curves) for the weighted boosting algorithm (solid curves) and the un-weighted algorithm (dashed curves). (b – right). Features $\mathbf{v}_t$ found by the learning algorithm.

estimated data distribution. To correct for this we include importance weights $g_n$ on the negative examples that are all $1/N$ at $\alpha_t = 0$. It is very easy to update these weights from iteration to iteration using $g_n \rightarrow g_n \exp(-\delta\alpha_t \; \phi_t(\mathbf{s}_n))$ and renormalizing. It is well known that in high dimensions the effective sample size of the weighted sample can rapidly become too small to be useful. We therefore monitor the effective sample size, given by $1/\sum_n g_n^2$, where the sum runs over the negative examples only. If it drops below a threshold we have two choices. We can obtain a new set of negative examples from the updated Boltzmann distribution, reset the importance weights to $1/N$ and resume fitting $\alpha_t$. Alternatively, we simply accept the current value of $\alpha_t$ and proceed to the next round of boosting. Because we initialize $\alpha_t = 0$ in the fitting procedure, the latter approach underestimates the importance of this particular feature, which is not a problem since a similar feature can be added in the next round.

## 4 A Binary Example: The Generalized RBM

We propose a simple extension of the "restricted Boltzmann machine" (RBM) with (+1,-1)-units [1] as a model for binary data. Each feature is parametrized by weights $\mathbf{v}_t$ and a bias $\beta_t$:

$$\alpha_t\phi_t(\mathbf{s}) = -\alpha_t \log \cosh(\mathbf{v}_t^T\mathbf{s} + \beta_t) \qquad (12)$$

where the RBM is obtained by setting all $\alpha_t = 1$. One can sample from the summed energy model using straightforward Gibbs sampling, where every visible unit is sampled given all the others. Alternatively, one can design a much faster mixing Markov chain by introducing hidden variables and sampling all hidden units independently given the visible units and vice versa. Unfortunately, by including the coefficients $\alpha_t$ this trick is no longer valid. But an approximate Markov chain can be used

$$\alpha_t \log \cosh(\mathbf{v}_t^T\mathbf{s} + \beta_t) \approx \log \cosh(\alpha_t\mathbf{v}_t\mathbf{s} + \alpha_t\beta_t) \qquad (13)$$

This approximate Gibbs sampling thus involves sampling from an RBM with scaled weights and biases,

$$P(h_t = 1|\mathbf{s}) = \sigma(2\alpha_t\mathbf{v}_t^T\mathbf{s} + 2\alpha_t\beta_t) \qquad P(s_i = 1|\mathbf{h}) = \sigma(2\sum_t \alpha_t v_{it} h_t) \qquad (14)$$

When using the above Markov chain, we will not wait until it has reached equilibrium but initialize it at the data-vectors and use it for a fixed number of steps, as is done in contrastive divergence learning [4].

When we fit a new feature we need to make sure its norm is controlled. The appropriate value depends on the number of dimensions in the problem; in the experiment described below we bounded the norm of the vector $[\mathbf{w}_t^T, \beta_t]$ to be no larger than $0.1$. The updates are thus given by $\mathbf{v}_t \rightarrow \mathbf{v}_t + \delta\mathbf{v}_t$ and $\beta_t \rightarrow \beta_t + \delta\beta_t$ with,

$$\delta\mathbf{v}_t \propto \sum_n w_n y_n \tanh(\mathbf{v}_t^T \mathbf{s}_n + \beta_t)\mathbf{s}_n \qquad \delta\beta_t \propto \sum_n w_n y_n \tanh(\mathbf{v}_t^T \mathbf{s}_n + \beta_t) \qquad (15)$$

where the weights $w_n$ are proportional to $\sigma(-y_n E_n)$. The coefficients $\alpha_t$ are determined using the procedure of Section 3.3.

To test whether we can learn good models of (fairly) high-dimensional, real-world data, we used the $16 \times 16$ real-valued digits from the "br" set on the CEDAR cdrom # 1. We learned completely separate models on binarized "2"s and "3"s. The first 600 data cases of each class were used for training while the remaining 500 digits of each class were used for testing. The minimum effective sample size for the coefficients $\alpha_t$ was set to $60\%$. We used 2 different sets of negative examples, 600 examples each, to fit $\phi_t(\cdot)$ and $\alpha_t$. After a new feature was added, the total energies of all "2"s and "3"s were computed under both models. The energies of the training data (under both models) were used as two-dimensional features to compute a separation boundary using logistic regression, which was subsequently applied to the test data to compute the total misclassification. In Figure 1a we show the total error on both training data and test data as a function of the number of features in the model. For comparison we also plot the training and test error for the un-weighted version of the algorithm ($w_n = 1, \forall\ n$). The classification error after 100 rounds of boosting for the weighted algorithm is about $0.65\%$, and only very gradually increases to about $0.75\%$ after 600 rounds of boosting. This is good as compared to logistic regression ($2.7\%$), k-nearest neighbors ($1.0\%$, $k = 1$ is optimal), while a parallel-trained RBM with $50, 100, 200$ hidden units achieves $0.9\%, 0.7\%, 0.85\%$ respectively. The un-weighted learning algorithm converges much more slowly to a good solution, both on training and test data. In Figure 1b we show every $10^{th}$ feature $\mathbf{v}_t$ between rounds 1 and 200 for both digits.

## 5   A Continuous Example: The Dimples Model

For continuous data we propose a different form of feature, which we term a *dimple* because of its shape in the energy domain. A dimple is a mixture of a narrow Gaussian and a broad Gaussian, with a common mean:

$$\phi_t(\mathbf{s}) = -\log\left[\mathcal{N}(\mathbf{s}; \mu, \sigma_1) + \mathcal{N}(\mathbf{s}; \mu, \sigma_2)\right] \qquad (16)$$

where the mixing proportion is constant and equal, and $\sigma_2$ is fixed and large. Each round of the algorithm fits $\mu$ and $\sigma_1$ for a new learner. A nice property of dimples is that they can reduce the entropy of an existing distribution by placing the dimple in a region that already has low energy, but they can also raise the entropy by putting the dimple in a high energy region [5].

Sampling is again simple if all $\alpha_t = 1$, since in that case we can use a Gibbs chain which first picks a narrow or broad Gaussian for every feature given the visible variables and then samples the visible variables from the resulting multivariate Gaussian. For general $\alpha$ the situation is less tractable, but using a similar approximation as for the generalized RBM,

$$\alpha\log\left[\mathcal{N}(\mathbf{s}; \mu, \sigma_1) + \mathcal{N}(\mathbf{s}; \mu, \sigma_2)\right] \approx \log\left[\mathcal{N}(\mathbf{s}; \mu, \sigma_1)^\alpha + \mathcal{N}(\mathbf{s}; \mu, \sigma_2)^\alpha\right] \qquad (17)$$

This approximation will be accurate when one Gaussian is dominating the other, i.e., when the responsibilities are close to zero and one. This is expected to be the case in high-dimensional applications. In the low-dimensional example discussed below we implemented a simple MCMC chain with isotropic, normal proposal density which was initiated at the data-points and run for a fixed number of steps.

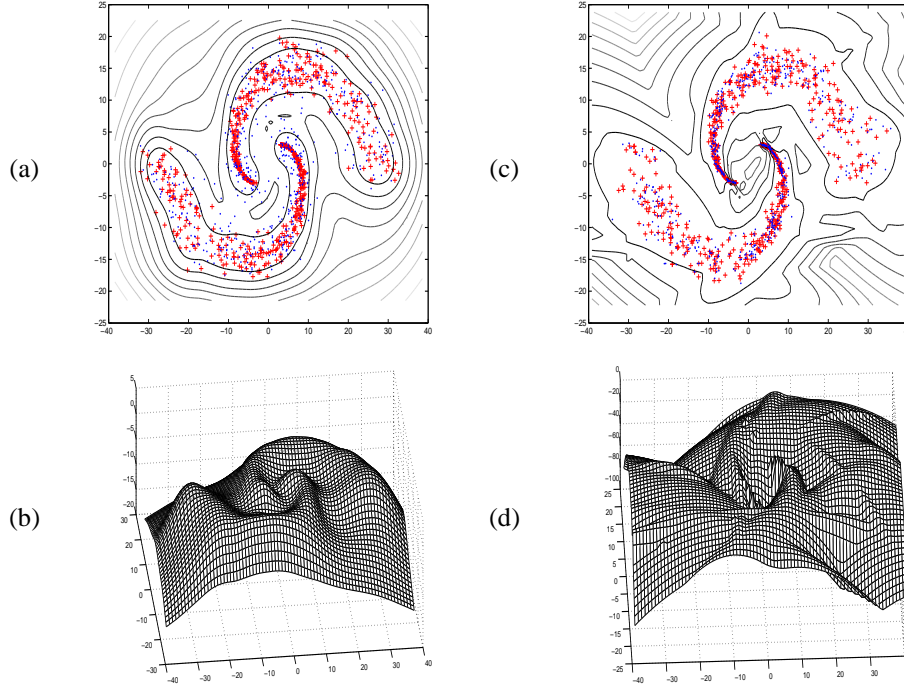

Figure 2: (a). Plot of iso-energy contours after 400 rounds of boosting. The crosses represent the data and the dots the negative examples generated from the model. (b). Three dimensional plot of the negative energy surface. (c). Contour plot for a mixture of 30 Gaussians learned using EM. (d). Negative energy surface for the mixture of 30 Gaussians.

The type of dimple we used in the experiment below can adapt a common mean ($\boldsymbol{\mu}$) and the inverse-variance of the small Gaussian ($\boldsymbol{\tau}_1$) in each dimension separately. The update rules are given by, $\boldsymbol{\mu} \to \boldsymbol{\mu} + \delta\boldsymbol{\mu}$ and $\boldsymbol{\tau}_1 \to \boldsymbol{\tau}_1 + \delta\boldsymbol{\tau}_1$ with

$$\delta\mu_i \propto \sum_n w_n y_n (X_{i,n} - \mu_i)(r_{1,n}\tau_{1,i} + r_{2,n}\tau_{2,i}) \tag{18}$$

$$\delta\tau_{1,i} \propto -\sum_n w_n y_n r_{1,n}\left((X_{i,n} - \mu_i)^2 - 1/\tau_{1,i}\right) \tag{19}$$

where $r_{1,n} = \mathcal{N}_1/(\mathcal{N}_1 + \mathcal{N}_2)$ and $r_{2,n} = 1 - r_{1,n}$ are the responsibilities for the narrow and broad Gaussian respectively and the weights are given by $w_n = \sigma(-y_n E_n)$. Finally, the combination coefficients $\boldsymbol{\alpha}_t$ are computed as described in Section 3.3.

To illustrate the proposed algorithm we fit the dimples model to the two-dimensional data (crosses) shown in Figure 2a-c. The data were synthetically generated by defining angles $\varphi_\pm = \pm\pi u$ with $u$ uniform between $[0, 1]$ and a radius $r_\pm = 10 + n\varphi_\pm$ with $n$ standard normal, which were converted to Euclidean coordinates and mirrored and translated to produce the spirals. The first feature is an isotropic Gaussian with the mean and the variance of the data, while later features were dimples trained in the way described above. Figure 2a also shows the contours of equal energy after 400 rounds of boosting together with examples (dots) from the model. A 3-dimensional plot of the negative energy surface is shown in Figure 2b. For comparison, similar plots for a mixture of 30 Gaussians, trained in parallel with EM, are depicted in Figures 2c and 2d.

The main qualitative difference between the fits in Figures 2a-b (product of dimples) and

2c-d (mixture of Gaussians), is that the first seems to produce smoother energy surfaces, only creating structure where there is structure in the data. This can be understood by recalling that the role of the negative examples is precisely to remove "dips" in the energy surface where there is no data. The philosophy of avoiding structure in the model that is not dictated by the data is consistent with the ideas behind maximum entropy modelling [11] and is thought to improve generalization.

## 6   Discussion

This paper discusses a boosting approach to density estimation, which we formulate as a sequential approach to training additive random field models. The philosophy is to view unsupervised learning as a sequence of classification problems where the aim is to discriminate between data-vectors and negative examples generated from the current model. The sampling step is usually the most time consuming operation, but it is also unavoidable since it informs the algorithm of the states whose energy is too low. The proposed algorithm uses just one sample of negative examples to fit a new feature, which is very economical as compared to most non-sequential algorithms which must generate an entire new sample for every gradient update.

There are many interesting issues and variations that we have not addressed in this paper. What is the effect of using approximate, e.g. variational distributions for $P(\mathbf{s})$? Can we improve the accuracy of the model by fitting the feature parameters and the coefficients $\alpha_t$ together? Does re-sampling the negative examples more frequently during learning improve the final model? What is the effect of using different functions to weight the data and how do the weighting schemes interact with the dimensionality of the problem?

## Footnotes

[1]Since $P(\mathbf{s})$ is independent of $\gamma_t$, it is easy to compute the second derivative $L_w'' = \sum_n y_n w_n'$ and we can do Newton updates to compute the stationary point.

## References

[1] Y. Freund and D. Haussler. Unsupervised learning of distributions of binary vectors using 2-layer networks. In *Advances in Neural Information Processing Systems*, volume 4, pages 912–919, 1992.

[2] J. Friedman, T. Hastie, and R. Tibshirani. Additive logistic regression: A statistical view of boosting. Technical report, Dept. of Statistics, Stanford University Technical Report., 1998.

[3] J.H. Friedman. Greedy function approximation: A gradient boosting machine. Technical report, Technical Report, Dept. of Statistics, Stanford University, 1999.

[4] G.E. Hinton. Training products of experts by minimizing contrastive divergence. *Neural Computation*, 14:1771–1800, 2002.

[5] G.E. Hinton and A. Brown. Spiking Boltzmann machines. In *Advances in Neural Information Processing Systems*, volume 12, 2000.

[6] G. Lebanon and J. Lafferty. Boosting and maximum likelihood for exponential models. In *Advances in Neural Information Processing Systems*, volume 14, 2002.

[7] L. Mason, J. Baxter, P. Bartlett, and M. Frean. Boosting algorithms as gradient descent. In *Advances in Neural Information Processing Systems*, volume 12, 2000.

[8] S. Della Pietra, V.J. Della Pietra, and J.D. Lafferty. Inducing features of random fields. *IEEE Transactions on Pattern Analysis and Machine Intelligence*, 19(4):380–393, 1997.

[9] S. Rosset and E. Segal. Boosting density estimation. In *Advances in Neural Information Processing Systems*, volume 15 (this volume), 2002.

[10] R.E. Schapire and Y. Singer. Improved boosting algorithms using confidence-rated predictions. In *Computational Learing Theory*, pages 80–91, 1998.

[11] S.C. Zhu, Z.N. Wu, and D. Mumford. Minimax entropy principle and its application to texture modeling. *Neural Computation*, 9(8):1627–1660, 1997.
